# Algorithms for Learning Markov Field Policies

**Abdeslam Boularias**
Max Planck Institute for Intelligent Systems
boularias@tuebingen.mpg.de

**Oliver Krömer, Jan Peters**
Technische Universität Darmstadt
{oli,jan}@robot-learning.de

## Abstract

We use a graphical model for representing policies in Markov Decision Processes. This new representation can easily incorporate domain knowledge in the form of a state similarity graph that loosely indicates which states are supposed to have similar optimal actions. A bias is then introduced into the policy search process by sampling policies from a distribution that assigns high probabilities to policies that agree with the provided state similarity graph, i.e. smoother policies. This distribution corresponds to a Markov Random Field. We also present forward and inverse reinforcement learning algorithms for learning such policy distributions. We illustrate the advantage of the proposed approach on two problems: cart-balancing with swing-up, and teaching a robot to grasp unknown objects.

## 1 Introduction

Markov Decision Processes (MDP) provide a rich and elegant mathematical framework for solving sequential decision-making problems. In practice, significant domain knowledge is often necessary for finding a near-optimal policy in a reasonable amount of time. For example, one needs a suitable set of basis functions, or features, to approximate the value functions in reinforcement learning and the reward functions in inverse reinforcement learning. Designing value or reward features can itself be a challenging problem. The features can be noisy, misspecified or insufficient, particularly in certain complex robotic tasks such as grasping and manipulating objects. In this type of applications, the features are mainly acquired through vision, which is inherently noisy. Many features are also nontrivial, such as the features related to the shape of an object, used for calculating grasp stability.

In this paper, we show how to overcome the difficult problem of designing precise value or reward features. We draw our inspiration from computer vision wherein similar problems have been efficiently solved using a family of graphical models known as *Markov Random Fields* (MRFs) (Kohli et al., 2007; Munoz et al., 2009). We start by specifying a graph that loosely indicates which pairs of states are supposed to have similar actions under an optimal policy. In an object manipulation task for example, the states correspond to the points of contact between the robot hand and the object surface. A state similarity graph can be created by sampling points on the surface of the object and connecting each point to its $k$ nearest neighbors using the geodesic or the Euclidean distance. The adjacency matrix of this graph can be interpreted as the *Gram matrix* of a kernel that can be used to approximate the optimal value function. Kernels have been widely used before in reinforcement learning (Ormoneit & Sen, 1999), however, they were used for approximating the values of different policies in a search for an optimal policy. Therefore, the kernels should span not only the optimal value function, but also the values of intermediate policies.

In this paper, kernels will be used for a different purpose. We only require that the kernel spans the value function of an optimal policy. Therefore, the value function of an optimal policy is assumed to have a low approximation error, measured by the *Bellman error*, using that kernel. Subsequently, we derive a distribution on policies, wherein the probability of a policy is proportional to its estimated value, and inversely proportional to its Bellman error. In other terms, the Bellman error is used as a surrogate function for measuring how close a policy is to an optimal one. We show that this

probability distribution is an MRF, and use a Markov chain Monte Carlo algorithm for sampling policies from it. We also describe an apprenticeship learning algorithm based on the same principal. A preliminary version of some parts of this work was presented in (Boularias et al., 2012).

## 2 Notations

Formally, a finite-horizon Markov Decision Process (MDP) is a tuple $(\mathcal{S}, \mathcal{A}, T, R, H, \gamma)$, where $\mathcal{S}$ is a set of states and $\mathcal{A}$ is a set of actions, $T$ is a transition function with $T(s, a, s') = P(s_{t+1} = s' | s_t = s, a_t = a)$ for $s, s' \in \mathcal{S}, a \in A$, and $R$ is a reward function where $R(s, a)$ is the reward given for action $a$ in state $s$. To ease notation and without loss of generality, we restrict our theoretical analysis to the case where rewards depend only on states, and denote by $R$ an $|\mathcal{S}| \times 1$ vector. $H$ is the planning horizon and $\gamma \in [0, 1]$ is a discount factor. A deterministic policy $\pi$ is a function that returns an action $a = \pi(s)$ for each state $s$. $T_\pi$ is defined as $T_\pi(s, s') = T(s, \pi(s), s')$. We denote by $\pi_{t:H}$ a non-stationary policy $(\pi_t, \pi_{t+1}, \ldots, \pi_H)$, where $\pi_i$ is a policy at time-step $i$. The value of policy $\pi_{t:H}$ is the expected sum of rewards received by following $\pi_{t:H}$, starting from a state $s$ at time $t$, $V_{\pi_{t:H}}(s) = \sum_{i=t}^{H} \gamma^{i-t} \mathbb{E}_{s_i}[R(s_i) | s_t = s, T_{\pi_{t:i}}]$. An optimal policy $\pi_{t:H}^*$ is one satisfying $\pi_{t:H}^* \in \arg\max_{\pi_{t:H}} V_{\pi_{t:H}}(s), \forall s \in \mathcal{S}$. Searching for an optimal policy is generally an iterative process with two phases: policy evaluation, and policy improvement.

When the state space $\mathcal{S}$ is large or continuous, the value function $V_{\pi_{t:H}}$ is approximated by a linear combination of $n$ basis functions, or features. Let $f_i$ be a $|\mathcal{S}||\mathcal{A}| \times 1$ vector corresponding to the $i^{\text{th}}$ basis function, and let $F$ be the $|\mathcal{S}||\mathcal{A}| \times n$ matrix of columns $f_i$. Let $\Pi_{\pi_t}$ be an $|\mathcal{S}| \times |\mathcal{S}||\mathcal{A}|$ action-selection matrix defined as $\Pi_{\pi_t}(s, (s, \pi_t(s))) = 1$ and 0 otherwise. Then $V_{\pi_{t:H}} = F_{\pi_t} w$, where $w$ is a $n \times 1$ weights vector and $F_{\pi_t} = \Pi_{\pi_t} F$. We define the *Bellman error* of two consecutive policies $\pi_t$ and $\pi_{t+1}$ using the feature matrix $F$ and the weights $w_t, w_{t+1} \in \mathbb{R}^n$ as $BE(F, w_{t:t+1}, \pi_{t:t+1}) = \|F_{\pi_t} w_t - \gamma T_{\pi_t} F_{\pi_{t+1}} w_{t+1} - R\|_1$. Similarly, we define the Bellman error of a distribution $P$ on policies $\pi_t$ and $\pi_{t+1}$ as $BE(F, w_{t:t+1}, P) = \|\mathbb{E}_{\pi_{t:t+1} \sim P}[F_{\pi_t} w_t - \gamma T_{\pi_t} F_{\pi_{t+1}} w_{t+1}] - R\|_1$. We also define the minimum Bellman error as $BE^*(F, \pi_{t:t+1}) = \min_{w_{t:t+1}} BE(F, w_{t:t+1}, \pi_{t:t+1})$ and the total Bellman error as $BE(F, w_{0:H}, \pi_{0:H}) = \sum_{t=0}^{H-1} BE(F, w_{t:t+1}, \pi_{t:t+1})$.

## 3 Markov Random Field Policies for Reinforcement Learning

We now present the reinforcement learning approach using the Bellman error as a structure penalty.

### 3.1 Structure penalty

Optimal policies of many real-world problems are structured and change smoothly over the state space. Therefore, the optimal value function can often be approximated by simple features, compared to the value functions of arbitrary policies. We exploit this property and propose to indirectly use these features, provided as domain knowledge, for accelerating the search for an optimal policy. Specifically, we restrain the policy search to a set of policies that have a low estimated Bellman error when their values are approximated using the provided features, knowing that the optimal policy has a low Bellman error. Note that our approach is complementary to function approximation methods. We only use the features for calculating Bellman errors, the value functions can be approximated by using other methods, such as LSTD (Boyan, 2002).

Let $K_\pi$ be the Gram matrix defined as $K_\pi = \Pi_\pi K \Pi_\pi^T$, where $K = FF^T$. Matrix $K$ is the adjacency matrix of a graph that indicates which states and actions are similar under an optimal policy. Feature matrix $F$ is not explicitly required, as only the matrix $K$ will be used later. Therefore, the user needs only to provide a similarity measure between states, such as the Euclidean distance.

Let $w_t, w_{t+1} \in \mathbb{R}^{|\mathcal{S}|}, \epsilon \in \mathbb{R}$, if $\|\mathbb{E}_{\pi_{t:t+1} \sim P}[K_{\pi_t} w_t - \gamma T_{\pi_t} K_{\pi_{t+1}} w_{t+1}] - R\|_1 \leq \epsilon$ then $BE^*(F, P) \leq \epsilon$. This result is obtained by setting $F^T \Pi_\pi^T w_t$ and $F^T \Pi_\pi^T w_{t+1}$ as the weight vectors of the values of policies $\pi_t$ and $\pi_{t+1}$. The condition above implies that the policy distribution $P$ has a value function that can be approximated by using $F$. Enforcing this condition results in a bias favoring policies with a low Bellman error. Thus, we are interested in learning a distribution $P(\pi_{0:H})$ that satisfies this condition, while maximizing its expected value.

Distribution $P$ can be decomposed using the chain rule as $P(\pi_{0:H}) = P(\pi_H)\prod_{t=0}^{H-1} P(\pi_t|\pi_{t+1:H})$. We start by calculating a distribution over deterministic policies $\pi_H$ that will be executed at the last time-step $H$. Then, for each step $t \in \{H-1, \ldots, 0\}$, we calculate a distribution $P(\pi_t|\pi_{t+1:H})$ over deterministic policies $\pi_t$ given policies $\pi_{t+1:H}$ that we sample from $P(\pi_{t+1:H})$. In the following, we show how to calculate $P(\pi_t|\pi_{t+1:H})$.

## 3.2 Primal problem

Let $\rho \in \mathbb{R}$ be a lower bound on the entropy of a distribution $P$ on deterministic policies $\pi_t$, conditioned on $\pi_{t+1:H}$. $\rho$ is used for tuning the exploration. Our problem can then be formulated as

$$\max_P \Big(\sum_{s\in\mathcal{S}} \mathbb{E}_P[V^{\pi_{t:H}}](s)\Big), \text{subject to}\Big(g_1(P)=1, g_2(P)\geq\rho, \|g_3(P)-R\|_1\leq\epsilon\Big), \tag{1}$$

where $\Big(g_1(P) = \sum_{\pi_t\in\mathcal{A}^{|\mathcal{S}|}} P(\pi_t|\pi_{t+1:H})\Big)$, $\Big(g_2(P) = -\sum_{\pi_t\in\mathcal{A}^{|\mathcal{S}|}} P(\pi_t|\pi_{t+1:H})\log P(\pi_t|\pi_{t+1:H})\Big)$,

$\Big(g_3(P) = \sum_{\pi_t\in\mathcal{A}^{|\mathcal{S}|}} P(\pi_t|\pi_{t+1:H})[K_{\pi_t}w_t - \gamma T_{\pi_t}K_{\pi_{t+1}}w_{t+1}]\Big)$, $\Big(\mathbb{E}_P[V^{\pi_{t:H}}] = \sum_{\pi_t\in\mathcal{A}^{|\mathcal{S}|}} P(\pi_t|\pi_{t+1:H})V^{\pi_{t:H}}\Big)$.

The objective function in Equation 1 is linear and its constraints define a convex set. Therefore, the optimal solution to Problem 1 can be found by solving its Lagrangian dual.

## 3.3 Dual problem

The Lagrangian dual is given by

$$L(P,\tau,\eta,\lambda) = \Big(\sum_{s\in\mathcal{S}} \mathbb{E}_P[V^{\pi_{t:H}}](s)\Big) - \eta\Big(g_1(P)-1\Big) + \tau\Big(g_2(P)-\rho\Big) + \lambda^T\Big(g_3(P)-R\Big) + \epsilon\|\lambda\|_1,$$

where $\eta, \tau \in \mathbb{R}$ and $\lambda \in \mathbb{R}^{|\mathcal{S}|}$. We refer the reader to Dudik et al. (2004) for a detailed derivation.

$$\frac{\partial L(P,\tau,\eta,\lambda)}{\partial P(\pi_t|\pi_{t+1:H})} = \sum_{s\in\mathcal{S}} V^{\pi_{t:H}}(s) + \lambda^T[K_{\pi_t}w_t - \gamma T_{\pi_t}K_{\pi_{t+1}}w_{t+1}] - \tau\log P(\pi_t|\pi_{t+1:H})) - \eta - 1.$$

By setting $\frac{\partial L(P,\tau,\eta,\lambda)}{\partial P(\pi_t|\pi_{t+1:H})} = 0$ (Karush-Kuhn-Tucker condition), we get the solution

$$P(\pi_t|\pi_{t+1:H}) \propto \exp\Big(\underbrace{\frac{1}{\tau}}_{\text{exploration factor}} \big(\overbrace{\sum_{s\in\mathcal{S}} V^{\pi_{t:H}}(s)}^{\text{expected sum of rewards}} + \underbrace{\lambda^T[K_{\pi_t}\mathbf{w}_t - \gamma T_{\pi_t}K_{\pi_{t+1}}\mathbf{w}_{t+1}]}_{\text{smoothness term}}\big)\Big).$$

This distribution on joint actions is a Markov Random Field. In fact, the kernel $K = FF^T$ is the adjacency matrix of a graph $(\mathcal{E},\mathcal{S})$, where $(s_i,s_j) \in \mathcal{E}$ if and only if $\exists a_i, a_j \in \mathcal{A} : K((s_i,a_i),(s_j,a_j)) \neq 0$. Local Markov property is verified, $\forall s_i \in \mathcal{S}$ : $P(\pi_t(s_i)|\pi_{t+1:H}, \{\pi_t(s_j) : s_j \in \mathcal{S}, s_j \neq s_i\}) = P(\pi_t(s_i)|\pi_{t+1:H}, \{\pi_t(s_j) : (s_i,s_j) \in \mathcal{E}, s_j \neq s_i\})$.

In other terms, the probability of selecting an action in a given state depends on the expected long term reward of the action, as well as on the selected actions in the neighboring states. Dependencies between neighboring states are due to the smoothness term in the distribution.

## 3.4 Learning parameters

Our goal now is to learn the distribution $P$, which is parameterized by $\tau, \lambda, w_{t:t+1}$ and $V^{\pi_{t:H}}$. Given that the transition function $T$ is unknown, we use samples $\mathcal{D} = \{(s_t, a_t, r_t, s_{t+1})\}$ for approximating the gradients of the parameters and the value function $V^{\pi_{t:H}}$. We also restrain $K_{\pi_t}$ to states and actions that appear in the samples, and denote by $\hat{T}_{\pi_t}$ the empirical transition matrix of the sampled states. Since $P(\pi_{0:H}) = P(\pi_H)\prod_{t=0}^{H-1} P(\pi_t|\pi_{t+1:H})$, then

$$P(\pi_{0:H}) \propto \exp\Big(\frac{1}{\tau}\sum_{t=0}^{H}\big(\sum_{s\in\mathcal{D}} V^{\pi_{t:H}}(s) + \lambda_t^T[K_{\pi_t}w_t - \gamma\hat{T}_{\pi_t}K_{\pi_{t+1}}w_{t+1}]\big)\Big). \tag{2}$$

The value function $V^{\pi_{t:H}}$ is empirically calculated from the samples by using a standard value function approximation algorithm, such as LSTD (Boyan, 2002). Temperature $\tau$ determines the entropy of the distribution $P$, $\tau$ is initially set to a high value and gradually decreased over time as more samples are collected. One can use the same temperature for all time-steps within the same episode, or a different one for each step. Since the Lagrangian $L$ is convex, parameters $\lambda_t$ can be learned by a simple gradient descent. Algorithm 1 summarizes the principal steps of the proposed approach. The algorithm iterates between two main steps: ($i$) sampling and executing policies from Equation 2, and ($ii$), updating the value functions and the parameters $\lambda_t$ using the samples. The weight vectors $w_{0:H}$ are the ones that minimize the empirical Bellman error in samples $\mathcal{D}$, they are also found by a gradient descent , wherein $\partial_{w_{0:H}} BE(K, w_{0:H}, \pi_{0:H})$ is estimated from $\mathcal{D}$.

---

**Algorithm 1** Episodic Policy Search with Markov Random Fields

---

Initialize the temperature $\tau$ with a large value, and $\lambda_{0:H}$ with 0.
**repeat**
   1. Sample policies $\pi_{0:H}$ from $P$ (Equation 2).
   2. Discard policies $\pi_{0:H}$ that have an empirical Bellman error higher than $\epsilon$.
   3. Execute $\pi_{0:H}$ and collect $\mathcal{D} = \{(s_t, a_t, r_t, s_{t+1})\}$.
   4. Update the value functions $V^{\pi_{t:H}}$ by using LSTD with $\mathcal{D}$.
   5. Find $\lambda_{0:H}$ that minimizes the dual $L$ by a gradient descent, $\partial_\lambda L$ is estimated from $\mathcal{D}$.
   6. Decrease the temperature $\tau$.
**until** $\tau \leq \epsilon_\tau$

---

The main assumption behind this algorithm is that the kernel $K$ approximates sufficiently well the optimal value function, what happens when this is not the case? The introduced bias will favor suboptimal policies. However, this problem can be solved by setting the threshold $\epsilon$ to a high value when the user is uncertain about the domain knowledge provided by $K$. Our experiments confirm that even a binary matrix $K$, corresponding to a $k$-NN graph, can yield an improved performance.

This approach is straightforward to extend to handle samples of continuous states and actions , in which case, a policy is represented by a vector $\Theta_t \in \mathbb{R}^N$ of continuous parameters (for instance, the center and the width of a gaussian). Therefore, Equation 2 defines a distribution $P(\Theta_{0:H})$. In our experiments, we use the Metropolis-Hastings algorithm for sampling $\Theta_{0:H}$ from $P$.

## 4 Markov Random Field Policies for Apprenticeship Learning

We now derive a policy shaping approach for apprenticeship learning using Markov Random Fields.

### 4.1 Apprenticeship learning

The aim of *apprenticeship learning* is to find a policy $\pi$ that is nearly as good as a policy $\hat{\pi}$ demonstrated by an expert, i.e., $V_\pi(s) \geq V_{\hat{\pi}}(s) - \epsilon, \forall s \in \mathcal{S}$. Abbeel & Ng (2004) proposed to learn a reward function, assuming that the expert is optimal, and to use it to recover the expert's generalized policy. The process of learning a reward function is known as *inverse reinforcement learning*. The reward function is assumed to be a linear combination of $m$ feature vectors $\phi_k$ with weights $\theta_k$, $\forall s \in \mathcal{S} : R(s) = \sum_{k=1}^m \theta_k \phi_k(s)$. The expected discounted sum of feature $\phi_k$, given policy $\pi_{t:H}$ and starting from $s$, is defined as $\phi_k^{\pi_{t:H}}(s) = \sum_{i=t}^H \gamma^{i-t} \mathbb{E}_{s_{t:H}}[\phi_k(s_i)|s_t = s, T_{\pi_{t:i}}]$. Using this definition, the expected return of a policy $\pi$ can be written as a linear function of the feature expectations, $V_{\pi_{t:H}}(s) = \sum_{k=1}^m \theta_k \phi_k^{\pi_{t:H}}(s)$. Since this problem is ill-posed, Ziebart et al. (2008) proposed to use the maximum entropy regularization, while matching the expected return of the examples. This latter constraint can be satisfied by ensuring that $\forall k, s : \phi_k^\pi(s) = \hat{\phi}_k$, where $\hat{\phi}_k$ denotes the empirical expectation of feature $\phi_k$ calculated from the demonstration.

### 4.2 Structure matching

The classical framework of apprenticeship learning is based on designing features $\phi$ of the reward and learning corresponding weights $\theta$. In practice, as we show in the experiments, it is often difficult to find an appropriate set of reward features. Moreover, the values of the reward features are usually

obtained from empirical data and are subject to measurement errors. However, most real-world problems exhibit a structure wherein states that are close together tend to have the same optimal action. This information about the structure of the expert's policy can be used to partially overcome the problem of finding reward features. The structure is given by a kernel that measures similarities between states. Given an expert's policy $\hat{\pi}_{0:H}$ and feature matrix $F$, we are interested in finding a distribution $P$ on policies $\pi_{0:H}$ that has a Bellman error similar to that of the expert's policy. The following proposition states the sufficient conditions for solving this problem.

**Proposition 1.** *Let $F$ be a feature matrix, $K = FF^T$, $K_{\pi_t} = \Pi_{\pi_t} K \Pi_{\pi_t}^T$. Let $P$ be a distribution on policies $\pi_t$ and $\pi_{t+1}$ such that $\mathbb{E}_{\pi_{t:t+1} \sim P}[K_{\pi_t}] = K_{\hat{\pi}_t}$, and $\mathbb{E}_{\pi_{t:t+1} \sim P}[\gamma T_{\pi_t} K_{\pi_{t+1}} T_{\pi_t}^T] = \gamma T_{\hat{\pi}_t} K_{\hat{\pi}_{t+1}} T_{\hat{\pi}_t}^T$, then $BE^*(F, \hat{\pi}_{t:t+1}) = BE^*(F, P)$.*

*Proof.* We prove that $BE^*(F, P) \leq BE^*(F, \hat{\pi}_{t:t+1})$. The same argument can be used for proving that $BE^*(F, \hat{\pi}_{t:t+1}) \leq BE^*(F, P)$. This proof borrows the orthogonality technique used for proving the Representer Theorem (Schölkopf et al., 2001). Let $\hat{w}_t, \hat{w}_{t+1} \in \mathbb{R}^{|\mathcal{S}|}$ be the weight vectors that minimize the Bellman error of the expert's policy, i.e. $\|\Pi_{\hat{\pi}_t} F \hat{w}_t - \gamma T_{\hat{\pi}_t} \Pi_{\hat{\pi}_{t+1}} F \hat{w}_{t+1} - R\|_p = BE^*(F, \hat{\pi}_{t:t+1})$. Let us write $\hat{w}_t = \hat{w}_{t\|} + \hat{w}_{t\perp}$, where $\hat{w}_{t\|}$ is the projection of $\hat{w}_t$ on the rows of $\Pi_{\hat{\pi}_t} F$, i.e. $\exists \hat{\alpha}_t \in \mathbb{R}^{|\mathcal{S}|} : \hat{w}_{t\|} = F^T \Pi_{\hat{\pi}_t}^T \hat{\alpha}_t$, and $\hat{w}_{t\perp}$ is orthogonal to the rows of $\Pi_{\hat{\pi}_t} F$. Thus, $\Pi_{\hat{\pi}_t} F \hat{w}_t = \Pi_{\hat{\pi}_t} F(\hat{w}_{t\|} + \hat{w}_{t\perp}) = \Pi_{\hat{\pi}_t} F \hat{w}_{t\|} = K_{\hat{\pi}_t} \hat{\alpha}_t$. Similarly, one can show that $\gamma T_{\hat{\pi}_t} \Pi_{\hat{\pi}_{t+1}} F \hat{w}_{t+1} = \gamma T_{\hat{\pi}_t} K_{\hat{\pi}_{t+1}} T_{\hat{\pi}_t}^T \hat{\alpha}_{t+1}$. Let $w_t = F^T \Pi_{\pi_t}^T \hat{\alpha}_t$ and $w_{t+1} = F^T \Pi_{\pi_{t+1}}^T T_{\pi_t}^T \hat{\alpha}_{t+1}$, then we have $BE^*(F, P) \leq \|\mathbb{E}_{\pi_{t:t+1}}[\Pi_{\pi_t} F w_t - \gamma T_{\pi_t} \Pi_{\pi_{t+1}} F w_{t+1}] - R\|_1 = \|\mathbb{E}_{\pi_{t:t+1}}[K_{\pi_t} \hat{\alpha}_t - \gamma T_{\pi_t} K_{\pi_{t+1}} T_{\pi_t}^T \hat{\alpha}_{t+1}] - R\|_1 = \|K_{\hat{\pi}_t} \alpha_{\hat{\pi}_t}^T - \gamma T_{\hat{\pi}_t} K_{\hat{\pi}_{t+1}} T_{\hat{\pi}_t}^T \alpha_{\hat{\pi}_{t+1}}^T - R\|_1 = BE^*(F, \hat{\pi}_{t:t+1})$. $\qquad \square$

### 4.3 Problem statement

Our problem now is to find a distribution on deterministic policies $P$ that satisfies the conditions stated in Proposition 1 in addition to the feature matching conditions $\phi_k^\pi(s) = \hat{\phi}_k$. The conditions of Proposition 1 ensure that $P$ assigns high probabilities to policies that have a structure similar to the expert's policy $\hat{\pi}$. The feature matching constraints ensure that the expected value under $P$ is the same as the value of the expert's policy. Given that there are infinite solutions to this problem, we select a distribution $P$ that has a maximal entropy (Ziebart et al., 2008).

$$\max_P \Big( \sum_{\pi_t \in \mathcal{A}^{|\mathcal{S}|}} -P(\pi_t|\pi_{t+1:H}) \log P(\pi_t|\pi_{t+1:H}) \Big),$$

$$\text{subject to} \Big( \sum_{\pi_t \in \mathcal{A}^{|\mathcal{S}|}} P(\pi_t|\pi_{t+1:H}) = 1 \Big), \Big( \sum_{\pi_t \in \mathcal{A}^{|\mathcal{S}|}} P(\pi_t|\pi_{t+1:H}) \phi^{\pi_t:H} = \hat{\phi} \Big),$$

$$\Big( \sum_{\pi_t \in \mathcal{A}^{|\mathcal{S}|}} P(\pi_t|\pi_{t+1:H}) K_{\pi_t} = K_{\hat{\pi}_t} \Big), \Big( \gamma T_{\hat{\pi}_t} K_{\hat{\pi}_{t+1}} T_{\hat{\pi}_t}^T = \sum_{\pi_t \in \mathcal{A}^{|\mathcal{S}|}} P(\pi_t|\pi_{t+1:H}) \gamma T_{\pi_t} K_{\pi_{t+1}} T_{\pi_t}^T \Big).$$

where $\phi^{\pi_t:H}(s, k) \overset{def}{=} \phi_k^{\pi_t:H}(s)$ (defined in subsection 4.1). The objective function of this problem is concave and the constraints are linear. Note that the three last equalities are between matrices.

### 4.4 Solution

By setting the derivatives of the Lagrangian to zero (as in subsection 3.3), we derive the distribution

$$P(\pi_t|\pi_{t+1:H}) \propto \exp\Big( \sum_k \sum_{s \in \mathcal{S}} \theta_k^s \phi_k^{\pi_t:H}(s) + \sum_{(s_i, s_j) \in \mathcal{S}^2} \lambda_{i,j} K_{\pi_t}(s_i, s_j) + \gamma \sum_{(s_i, s_j) \in \mathcal{S}^2} \xi_{i,j} (T_{\pi_t} K_{\pi_{t+1}} T_{\pi_t}^T)(s_i, s_j) \Big).$$

Again, this distribution is a Markov Random Field. The parameters $\theta$, $\lambda$ and $\xi$ are learned by maximizing the likelihood $P(\hat{\pi}_{t:H})$ of the expert's policy $\hat{\pi}_{t:H}$. The learned parameters can then be used for sampling policies that have the same expected value (from the second constraint), and the same Bellman error (from the last two constraints and Proposition 1) as the expert's policy. If kernel $K$ is inaccurate, then the learned $\lambda$ and $\xi$ will take low values to maximize the likelihood of the expert's policy. Hence, our approach will be reduced to MaxEnt IRL (Ziebart et al., 2008).

For simplicity, we consider an approximate solution with fewer parameters in our experiments, where each $\theta_k^s$ is replaced by $\theta_k \in \mathbb{R}$. This simplification is based on the fact that the reward function is independent of the initial state. We also replace $\lambda_{i,j}$ by $\lambda \in \mathbb{R}$, and $\xi_{i,j}$ by $\xi \in \mathbb{R}$.

For a sparse matrix $K$, one can create a corresponding graph $(\mathcal{E}, \mathcal{S})$, where $(s_i, s_j) \in \mathcal{E}$ if and only if $\exists a_i, a_j \in \mathcal{A} : K((s_i, a_i), (s_j, a_j)) \neq 0$ or $\exists a_i, a_j \in \mathcal{A}, (s'_i, s'_j) \in \mathcal{E} : \gamma T(s_i, a_i, s'_i) T(s_j, a_j, s'_j) \neq 0$. Finally, the policy distribution can be rewritten as

$$P(\pi_t | \pi_{t+1:H}) \propto \exp \Big( \sum_{s \in \mathcal{S}} V_\theta^{\pi_{t:H}}(s) + \lambda \big( \sum_{(s_i, s_j) \in \mathcal{E}} K_{\pi_t}(s_i, s_j) + \gamma \xi \sum_{(s_i, s_j) \in \mathcal{E}} (T_{\pi_t} K_{\pi_{t+1}} T_{\pi_t}^T)(s_i, s_j)) \big) \Big), (3)$$

where $V_\theta^{\pi_{t:H}}(s) = \sum_k \theta_k \phi_k(s) + \gamma \sum_{s' \in \mathcal{S}} T_{\pi_t}(s, s') V_\theta^{\pi_{t+1:H}}(s')$.

The distribution given by Equation 3 is a Markov Random Field. The probability of choosing action $a$ in a given state $s$ depends on the expected value of $(s, a)$ and the actions chosen in neighboring states. There is a clear similarity between this distribution of joint actions and the distribution of joint labels in Associative Markov Networks (AMN) (Taskar, 2004). In fact, the proposed framework generalizes AMN to sequential decision making problems. Also, the MaxEnt method (Ziebart et al., 2008) can be derived from Equation 3 by setting $\lambda = 0$.

|  | $\lambda = 0$ | $\lambda \neq 0$ |
|---|---|---|
| $\gamma = 0$ | Logistic regression | AMN (Taskar, 2004) |
| $\gamma \neq 0$ | MaxEnt IRL (Ziebart et al., 2008) | AL-MRF |

Table 1: Relation between Apprenticeship Learning with MRFs (AL-MRF) and other methods.

## 4.5 Learning procedure

In the learning phase, Equation 3 is used for finding parameters $\theta, \lambda$ and $\xi$ that maximize the likelihood of the expert's policy $\hat{\pi}$. Since this likelihood function is concave, a global optimal can be found by using standard optimization methods, such as BFGS. A main drawback of our approach is the high computational cost of calculating the partition function of Equation 3, which is $\mathcal{O}(|\mathcal{A}|^{|\mathcal{S}|} |\mathcal{S}|^2)$. In practice, this problem can be addressed by using several possible tricks. For instance, we reuse the values calculated for a given policy $\pi$ as the initial values of all the policies that differ from $\pi$ in one state only. We also decompose the state space into a set of weakly connected components, and separately calculate the partition of each component. One can also use recent efficient learning techniques for MRFs, such as (Krähenbühl & Koltun, 2011).

## 4.6 Planning procedure

Algorithm 2 describes a dynamic programming procedure for finding a policy $(\pi_0^*, \pi_1^*, \ldots, \pi_H^*)$ that satisfies $\forall t \in [0, H] : \pi_t^* \in \arg\max_{\pi_t \in \mathcal{A}^{|\mathcal{S}|}} P(\pi_t | \pi_{t+1:H}^*)$. The planning problem is reduced to a sequence of inference problems in Markov Random Fields. The inference problem itself can also be efficiently solved using techniques such as graph min-cut (Boykov et al., 1999), $\alpha$-expansions and linear programming relaxation (Taskar, 2004). We use the $\alpha$-expansions for our experiments.

---

**Algorithm 2** Dynamic Programming for Markov Random Field Policies

---

$\forall (s, a) \in \mathcal{S} \times \mathcal{A} : Q^{H+1}(s, a) = 0$.
**for** $t = H : 0$ **do**
    1. $\forall (s, a) \in \mathcal{S} \times \mathcal{A} : Q^t(s, a) = \sum_k \theta_k \phi_k(s) + \gamma \sum_{s'} T(s, a, s') Q^{t+1}(s', \pi_{t+1}^*(s'))$
    2. Use an inference algorithm (such as the $\alpha$-expansions) in the MRF defined on the graph $(\mathcal{S}, \mathcal{E})$ to label states with actions: the cost of labeling $s$ with $a$ is $-Q^t(s, a)$ and the potential of $(s_i, a_i, s_j, a_j)$ is $\lambda \Big( K(s_i, a_i, s_j, a_j) + \gamma \xi \sum_{(s'_i, s'_j) \in \mathcal{E}} T(s_i, a_i, s'_i) T(s_j, a_j, s'_j) K_{\pi_{t+1}^*}(s'_i, s'_j) \Big)$.
    3. Denote by $\pi_t^*$ the labeling policy returned by the inference algorithm;
**end for**
Return the policy $\pi^* = (\pi_0^*, \pi_1^*, \ldots, \pi_H^*)$;

---

## 5 Experimental Results

We present experiments on two problems: learning to swing-up and balance an inverted pendulum on a cart, and learning to grasp unknown objects.

## 5.1 Swing-up cart-balancing

The simulated swing-up cart-balancing system (Figure 1) consists of a 6 kg cart running on a 2 m track and a freely-swinging 1 kg pendulum with mass attached to the cart with a 50 cm rod. The state of the system is the position and velocity of the cart $(x, \dot{x})$, as well as the angle and angular velocity of the pendulum $(\theta, \dot{\theta})$. An action $a \in \mathbb{R}$ is a horizontal force applied to the cart. The dynamics of the system are nonlinear. States and actions are continuous, but time is discretized to steps of $0.1$ s. The objective is to learn, in a series of $5s$ episodes, a policy that swings the pendulum up and balances it in the inverted position. Since the pendulum falls down after hitting one of the two track limits, the policy should also learn to maintain the cart in the middle of the track. Moreover, the track has a nonuniform friction modeled as a force slowing down the cart. Part of the track has a friction of 30 N, while the remaining part has no friction. This variant is more difficult than the standard ones (Deisenroth & Rasmussen, 2011).

We consider parametric policies of the form $\pi(x, \dot{x}, \theta, \dot{\theta}) = \sum_i p_i q_i(x, \dot{x}, \theta, \dot{\theta})$, where $p_i$ are real weights and $q_i$ are basis functions corresponding to the signs of the angle and the angular velocity and an exponential function centered at the middle of the track. Moreover, we discretize the track into 10 segments, and use 10 binary basis functions for friction compensation, each one is nonzero only in a particular segment. A reward of 1 is given for each step the pendulum is above the horizon.

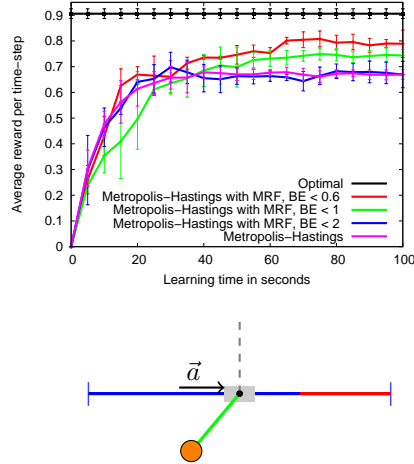

Since the friction changes smoothly along the track (domain knowledge), we use the adjacency matrix of a nearest-neighbor graph as the MRF kernel $K$ in Equation 2. Specifically, we set $K\big(\langle x_i, \dot{x}_i, \theta_i, \dot{\theta}_i, u_i \rangle, \langle x_j, \dot{x}_j, \theta_j, \dot{\theta}_j, u_j \rangle\big) = 1$ iff $|x_i - x_j| \leq 0.2m$, $\theta_i \theta_j \geq 0$, $\dot{\theta}_i \dot{\theta}_j \geq 0$, and $|u_i - u_j| \leq 5N$, otherwise $K$ is set to 0. Figure 1 shows the average reward per time-step of the learned policies as a function of the learning time. Our attempts to solve this variant using different policy gradient methods, e.g. (Kober & Peters, 2008), mainly resulted in poor policies. We report the values of the policies sampled with Metropolis-Hastings using Equation 2, and compare to the case where the policies are sampled solely according to their expected values, i.e. $\lambda_t = 0$. The expected values are estimated from the samples. The results, averaged over 50 independent trials, show that the convergence is faster when the MRF is used. Moreover, the performance increases as the threshold set on the maximum Bellman error ($\epsilon$) in Algorithm 1 is decreased. In fact, policies that change smoothly have a lower Bellman error as their values can be better approximated with kernel $K$.

Figure 1: Swing-up cart-balancing. The friction is nonuniform, the red area has a higher friction than the blue one. However, the friction changes only at one point of the track. Consequently, restraining the search to smooth policies yield faster convergence.

## 5.2 Precision grasps of unknown objects

From a high-level point of view, grasping an object can be seen as an MDP with three steps: reaching, preshaping, and grasping. At any step, the robot can either proceed to the next step or restart from the beginning and get a reward of 0. At $t = 0$, the robot always starts from the same initial state $s_0$, and the set of actions corresponds to the set of points on the surface of the object. Given a grasping point, we set the approach direction to the surface normal vector. At $t = 1$, the state is given by a surface point and an approach direction, and the set of actions corresponds to the set of all possible hand orientations. At $t = 2$, the state is given by a surface point, an approach direction and a hand orientation. There are two possible last actions, closing the fingers or restarting.

In this experiment, we are interested in learning to grasp objects from their handles. The reward of each step depends on the current state. There is no reward at $t = 0$. The reward $R_1$ defined at $t = 1$ is a function of the first three eigenvalues of the scatter matrix defined by the 3D coordinates of the points inside a small ball centered on the selected point (Boularias et al., 2011). The reward $R_2$,

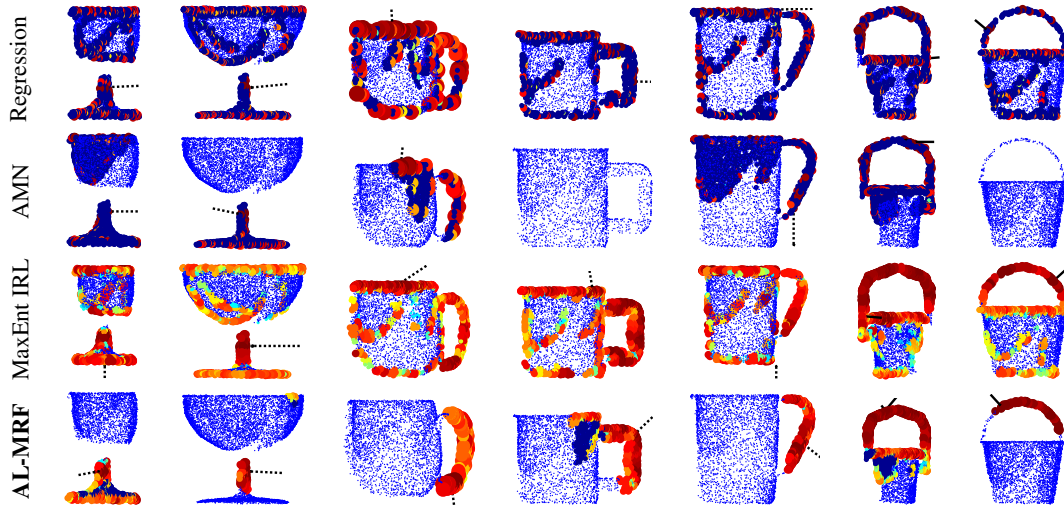

Table 2: Learned Q-values at $t = 0$. Each point on an object corresponds to a reaching action. Blue indicates low values and red indicates high values. The black arrow indicates the approach direction in the optimal policy according to the learned reward function.

defined at $t = 2$, is a function of collision features. We simulate the trajectories of 10 equidistant points on each finger of a Barrett hand (a three-fingered gripper). The collision features are binary variables indicating whether or not the corresponding finger points will make contact with the object.

Based on the domain knowledge that points that are close to each other should have the same action (i.e. same approach direction and hand orientation), the kernel $K$ is given by the $k$-nearest neighbors graph, using the Euclidean distance and $k = 6$ in the state space of positions (or surface points), and the angular distance, with $k = 2$ in the discretized state space of hand orientations. We also use a quadratic kernel for learning $R_1$, and the Hamming distance between the feature vectors as a kernel for learning $R_2$. We also use a single constant feature for all the edges.

We used one object for training and provided six trajectories leading to a successful grasp from its handle. For testing, we compared our approach (Apprenticeship Learning with MRF) with MaxEnt IRL, AMN and Logistic Regression, which is equivalent to AMN without the graph structure. For AMN and Logistic Regression, only the reward $R_1$ at time-step 1 is learned, since these are classification methods and do not consider subsequent rewards.

Table 2 shows the Q-values at $t = 0$ and the approach directions at optimal grasping points. AL-MRF improves over the other methods by generally giving high values to handle points only. The values of the other points are zeros because the optimal action at these points is to restart rather than to grasp. The confusion in the other methods comes from noised point coordinates and self-occlusions. More importantly, AL-MRF improves over AMN, a structured supervised learning technique, by considering the reward at $t = 2$ while making a decision at $t = 1$. This can be seen as a type of object recognition by functionality. Figure 2

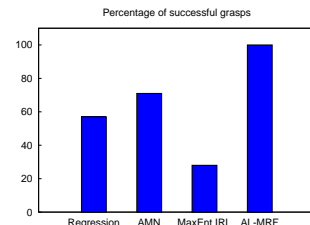

Figure 2: Percentage of grasps located on a handle with a correct approach direction and hand orientation.

shows the percentage of successful grasps using the objects in Table 2. A grasp is successful if it is located on a handle and the hand orientation is orthogonal to the handle and the approach direction.

## 6 Conclusion

Based on the observation that the value function of an optimal policy is often smooth and can be approximated with a simple kernel, we introduced a general framework for incorporating this type of domain knowledge in forward and inverse reinforcement learning. Our approach uses Markov Random Fields for defining distributions on deterministic policies, and assigns high probabilities to smooth policies. We also provided strong empirical evidence of the advantage of this approach.

**Acknowledgement**
This work was partly supported by the EU-FP7 grant 248273 (GeRT).

# References

Abbeel, Pieter and Ng, Andrew Y. Apprenticeship Learning via Inverse Reinforcement Learning. In *Proceedings of the Twenty-first International Conference on Machine Learning (ICML'04)*, pp. 1–8, 2004.

Boularias, Abdeslam, Krömer, Oliver, and Peters, Jan. Learning robot grasping from 3-D images with Markov Random Fields. In *Proceedings of the 2011 IEEE/RSJ International Conference on Intelligent Robots and Systems (IROS'11)*, pp. 1548–1553, 2011.

Boularias, Abdeslam, Krömer, Oliver, and Peters, Jan. Structured Apprenticeship Learning. In *Proceedings of the European Conference on Machine Learning and Knowledge Discovery in Databases (ECML-PKDD'12)*, pp. 227–242, 2012.

Boyan, Justin A. Technical Update: Least-Squares Temporal Difference Learning. *Machine Learning*, 49:233–246, November 2002. ISSN 0885-6125.

Boykov, Yuri, Veksler, Olga, and Zabih, Ramin. Fast Approximate Energy Minimization via Graph Cuts. *IEEE Transactions on Pattern Analysis and Machine Intelligence*, 23:2001, 1999.

Deisenroth, Marc Peter and Rasmussen, Carl Edward. PILCO: A Model-Based and Data-Efficient Approach to Policy Search. In *Proceedings of the Twenty-Eighth International Conference on Machine Learning (ICML'11)*, pp. 465–472, 2011.

Dudik, Miroslav, Phillips, Steven J., and Schapire, Robert E. Performance guarantees for regularized maximum entropy density estimation. In *Proceedings of the 17th Annual Conference on Computational Learning Theory (COLT'04)*, pp. 472–486, 2004.

Kober, Jens and Peters, Jan. Policy search for motor primitives in robotics. In *NIPS*, pp. 849–856, 2008.

Kohli, Pushmeet, Kumar, Pawan, and Torr, Philip. P3 and beyond: Solving energies with higher order cliques. In *Proceedings of IEEE Conference on Computer Vision and Pattern Recognition (CVPR'07)*, 2007.

Krähenbühl, Philipp and Koltun, Vladlen. Efficient Inference in Fully Connected CRFs with Gaussian Edge Potentials. In *Advances in Neural Information Processing Systems 24*, pp. 109–117. 2011.

Munoz, Daniel, Vandapel, Nicolas, and Hebert, Martial. Onboard contextual classification of 3-D point clouds with learned high-order Markov random fields. In *Proceedings of IEEE International Conference on Robotics and Automation (ICRA'09)*, 2009.

Ormoneit, Dirk and Sen, Saunak. Kernel-based reinforcement learning. In *Machine Learning*, pp. 161–178, 1999.

Schölkopf, Bernhard, Herbrich, Ralf, and Smola, Alex. A Generalized Representer Theorem . *Computational Learning Theory*, 2111:416–426, 2001.

Taskar, Ben. *Learning Structured Prediction Models: A Large Margin Approach*. PhD thesis, Stanford University, CA, USA, 2004.

Ziebart, B., Maas, A., Bagnell, A., and Dey, A. Maximum Entropy Inverse Reinforcement Learning. In *Proceedings of the Twenty-Second AAAI Conference on Artificial Intelligence (AAAI'08)*, pp. 1433–1438, 2008.

